# An *in-silico* Neural Model of Dynamic Routing through Neuronal Coherence

**Devarajan Sridharan**[*†]**, Brian Percival**[*‡]**, John Arthur**[♮] **and Kwabena Boahen**[♮]
[†] Program in Neurosciences,
[‡] Department of Electrical Engineering
and [♮] Department of Bioengineering
Stanford University
[*] These authors contributed equally
{dsridhar, bperci, jarthur, boahen}@stanford.edu

## Abstract

We describe a neurobiologically plausible model to implement dynamic routing using the concept of neuronal communication through neuronal coherence. The model has a three-tier architecture: a raw input tier, a routing control tier, and an invariant output tier. The correct mapping between input and output tiers is realized by an appropriate alignment of the phases of their respective background oscillations by the routing control units. We present an example architecture, implemented on a neuromorphic chip, that is able to achieve circular-shift invariance. A simple extension to our model can accomplish circular-shift dynamic routing with only $O(N)$ connections, compared to $O(N^2)$ connections required by traditional models.

## 1 Dynamic Routing Circuit Models for Circular-Shift Invariance

Dynamic routing circuit models are among the most prominent neural models for invariant recognition [1] (also see [2] for review). These models implement shift invariance by dynamically changing spatial connectivity to transform an object to a standard position or orientation. The connectivity between the raw input and invariant output layers is controlled by routing units, which turn certain subsets of connections on or off (Figure 1A). An important feature of this model is the explicit representation of *what* and *where* information in the main network and the routing units, respectively; the routing units use the *where* information to create invariant representations.

Traditional solutions for shift invariance are neurobiologically implausible for at least two reasons. First, there are too many synaptic connections: for $N$ input neurons, $N$ output neurons and $N$ possible input-output mappings, the network requires $O(N^2)$ connections in the routing layer—between each of the $N$ routing units and each set of $N$ connections that that routing unit gates (Figure 1A). Second, these connections must be extremely precise: each routing unit must activate an input-output mapping ($N$ individual connections) corresponding to the desired shift (as highlighted in Figure 1A). Other approaches that have been proposed, including *invariant feature networks* [3,4], also suffer from significant drawbacks, such as the inability to explicitly represent *where* information [2]. It remains an open question how biology could achieve shift invariance without profligate and precise connections.

In this article, we propose a simple solution for shift invariance for quantities that are circular or periodic in nature—circular-shift invariance (CSI)—orientation invariance in vision and key invariance in music. The visual system may create orientation-invariant representations to aid recognition under conditions of object rotation or head-tilt [5,6]; a similar mechanism could be employed by the auditory system to create key-invariant representations under conditions where the same melody

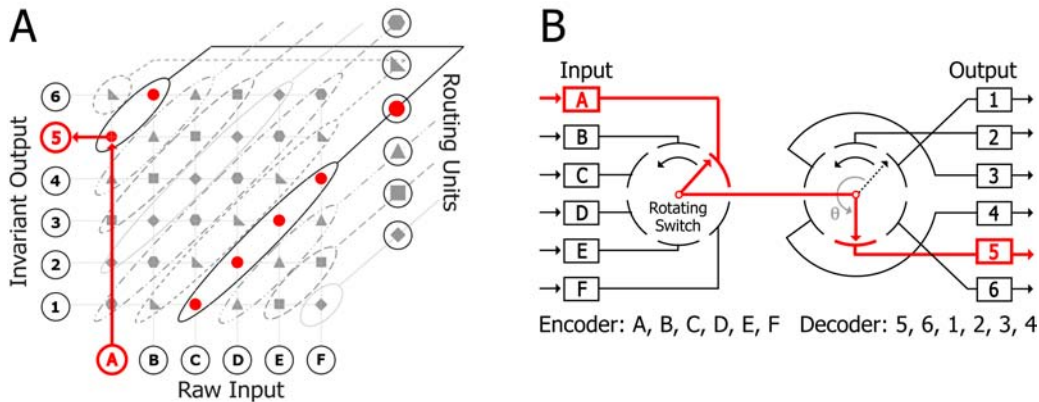

Figure 1: Dynamic routing. **A** In traditional dynamic routing, connections from the (raw) input layer to the (invariant) output layer are gated by routing units. For instance, the mapping from A to 5, B to 6, . . . , F to 4 is achieved by turning on the highlighted routing unit. **B** In time-division multiplexing (TDM), the encoder samples input channels periodically (using a rotating switch) while the decoder sends each sample to the appropriate output channel (based on its time bin). TDM can be extended to achieve a circular-shift transformation by altering the angle between encoder and decoder switches ($\theta$), thereby creating a rotated mapping between input and output channels (adapted from [7]).

is played in different keys. Similar to orientation, which is a periodic quantity, musical notes one octave apart sound alike, a phenomenon known as octave equivalence [8]. Thus, the problems of key invariance and orientation invariance admit similar solutions.

Deriving inspiration from time-division multiplexing (TDM), we propose a neural network for CSI that uses phase to encode and decode information. We modulate the temporal window of communication between (raw) input and (invariant) output neurons to achieve the appropriate input–output mapping. Extending TDM, any particular circular-shift transformation can be accomplished by changing the relative angle, $\theta$, between the rotating switches of the encoder (that encodes the raw input in time) and decoder (that decodes the invariant output in time) (Figure 1B). This obviates the need to hardwire routing control units that specifically modulate the strength of each possible input-output connection, thereby significantly reducing the complexity inherent in the traditional dynamic routing solution. Similarly, a remapping between the input and output neurons can be achieved by introducing a relative phase-shift in their background oscillations.

## 2 Dynamic Routing through Neuronal Coherence

To modulate the temporal window of communication, the model uses a ring of neurons (the *oscillation* ring) to select the pool of neurons (in the *projection* ring) that encode or decode information at a particular time (Figure 2A). Each projection pool encodes a specific value of the feature (for example, one of twelve musical notes). Upon activation by external input, each pool is active only when background inhibition generated by the oscillation ring (outer ring of neurons) is at a minimum. In addition to exciting 12 inhibitory interneurons in the projection ring, each oscillation ring neuron excites its nearest 18 neighbors in the clockwise direction around the oscillation ring. As a result, a wave of inhibition travels around the projection ring that allows only one pool to be excitable at any point in time. These neurons become excitable at roughly the same time (numbered sectors, inner ring) by virtue of recurrent excitatory intra-pool connections.

Decoding is accomplished by a second tier of rings (Figure 2B). The projection ring of the first (input) tier connects all-to-all to the projection ring of the second (output) tier. The two oscillation rings create a window of excitability for the pools of neurons in their respective projection rings. Hence, the most effective communication occurs between input and output pools that become excitable at the same time (i.e. are oscillating in phase with one another [9]).

The CSI problem is solved by introducing a phase-shift between the input and output tiers. If they are exactly in phase, then an input pool is simply mapped to the output pool directly above it. If their

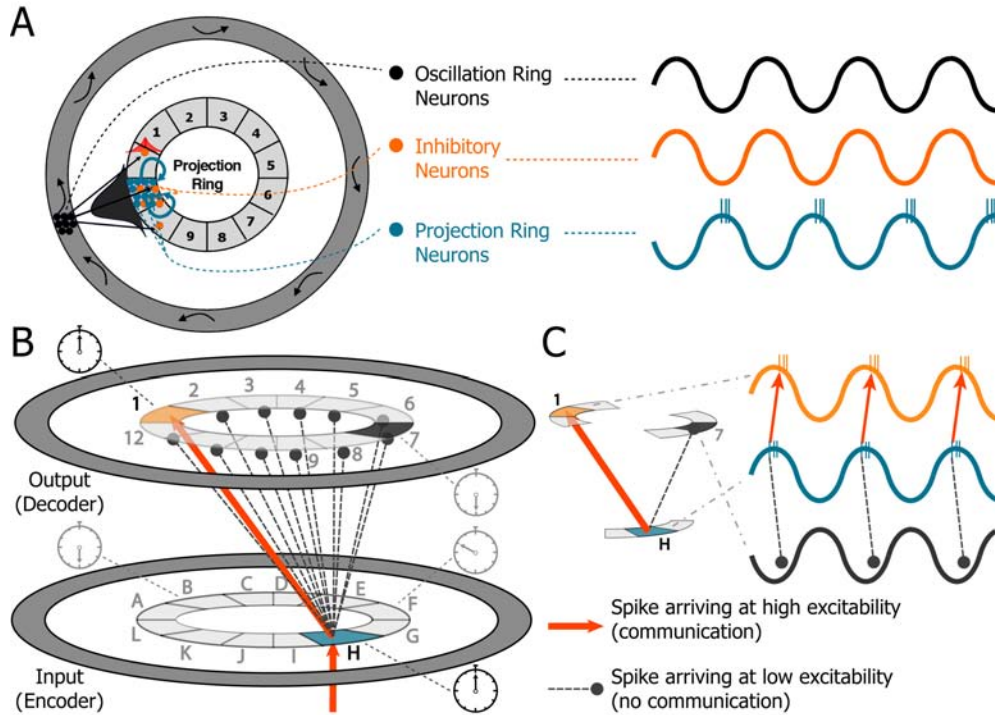

Figure 2: Double-Ring Network for Encoding and Decoding. **A** The projection (inner) ring is divided into (numbered) pools. The oscillation (outer) ring modulates sub-threshold activity (wave-forms) of the projection ring by exciting (black distribution) inhibitory neurons that inhibit neighboring projection neurons. A wave of activity travels around the oscillation ring due to asymmetric excitatory connections, creating a corresponding wave of inhibitory activity in the projection ring, such that only one pool of projection neurons is excitable (spikes) at a given time. **B** Two instances of the double-ring structure from A. The input projection ring connects all-to-all to the output projection ring (dashed lines). Because each input pool will spike only during a distinct time bin, and each output pool is excitable only in a certain time bin, communication occurs between input and output pools that are oscillating in phase with each other. Appropriate phase offset between input and output oscillation rings realizes the desired circular shift (input pool H to output pool 1, solid arrow). **C** Interactions among pools highlighted in B.

phases are different, the input is dynamically routed to an appropriate circularly shifted position in the output tier. Such changes in phase are analogous to adjusting the angle of the rotating switch at either the encoder or the decoder in TDM (see Figure 1B). There is some evidence that neural systems could employ phase relationships of subthreshold oscillations to selectively target neural populations [9-11].

# 3   Implementation in Silicon

We implemented this solution to CSI on a neuromorphic silicon chip [12]. The neuromorphic chip has neurons whose properties resemble that of biological neurons; these neurons even have intrinsic differences, thereby mimicking heterogeneity in real neurobiological systems. The chip uses a conductance-based spiking model for both inhibitory and excitatory neurons. Inhibitory neurons project to nearby excitatory and inhibitory neurons via a diffusor network that determines the spread of inhibition. A lookup table of excitatory synaptic connectivity is stored in a separate random-access memory (RAM) chip. Spikes occurring on-chip are converted to a neuron address, mapped to synapses (if any) via the lookup table, and routed to the targeted on-chip synapse. A universal serial bus (USB) interface chip communicates spikes to and from a computer, for external input and

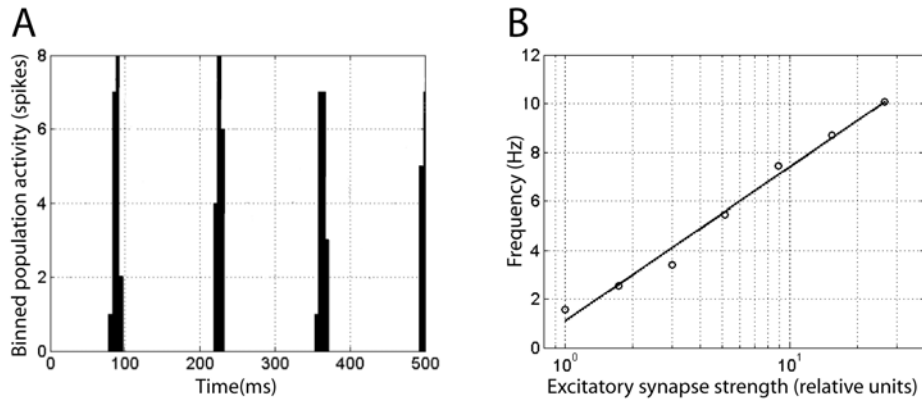

Figure 3: Traveling-wave activity in the oscillation ring. **A** Population activity (5ms bins) of a pool of eighteen (adjacent) oscillation neurons. **B** Increasing the strength of feedforward excitation led to increasing frequencies of periodic firing in the $\theta$ and $\alpha$ range (1-10 Hz). Strength of excitation is the amplitude change in post-synaptic conductance due to a single pre-synaptic spike (measured relative to minimum amplitude used).

data analysis, respectively. Simulations on the chip occur in real-time, making it an attractive option for implementing the model.

We configured the following parameters:

- Magnitude of a potassium M-current: increasing this current's magnitude increased the post-spike repolarization time of the membrane potential, thereby constraining spiking to a single time bin per cycle.

- The strength of excitatory and inhibitory synapses: a correct balance had to be established between excitation and inhibition to make only a small subset of neurons in the projection rings fire at a time—too much excitation led to widespread firing and too much inhibition led to neurons that were entirely silent or fired sporadically.

- The space constant of inhibitory spread: increasing the spread was effective in preventing runaway excitation, which could occur due to the recurrent excitatory connections.

We were able to create a stable traveling wave of background activity within the oscillation ring. We transiently stimulated a small subset of the neurons, which initiated a wave of activity that propagated in a stable manner around the ring after the transient external stimulation had ceased (Figure 3A). The network frequency determined from a Fourier transform of the network activity smoothed with a non-causal Gaussian kernel (FDHM = 80ms) was 7.4Hz. The frequency varied with the strength of the neurons' excitatory connections (Figure 3B), measured as the amplitude of the step increase in membrane conductivity due to the arrival of a pre-synaptic spike. Over much of the range of the synaptic strengths tested, we observed stable oscillations in the $\theta$ and $\alpha$ bands (1-10Hz); the frequency appeared to increase logarithmically with synaptic strength.

## 4 Phase-based Encoding and Decoding

In order to assess the best-case performance of the model, the background activity in the input and output projection rings was derived from the input oscillation ring. Their spikes were delivered to the appropriately circularly-shifted output oscillation neurons. The asymmetric feedforward connections were disabled in the output oscillation ring. For instance, in order to achieve a circular shift by $k$ pools (i.e. mapping input projection pool 1 to output projection pool $k + 1$, input pool 2 to output pool $k + 2$, and so on), activity from the input oscillation neurons closest to input pool 1 was fed into the output oscillation neurons closest to output pool $k$. By providing the appropriate phase difference between input and output oscillation, we were able to assess the performance of the model under ideal conditions. In the Discussion section, we discuss a biologically plausible mechanism to control the relative phases.

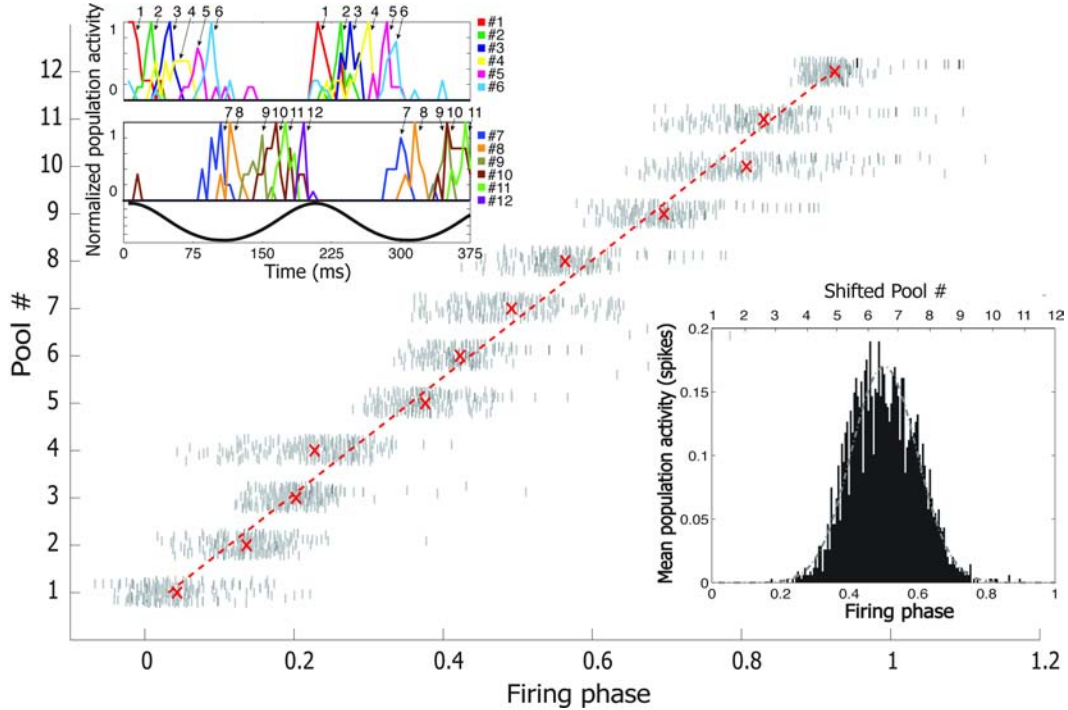

Figure 4: Phase-based encoding. Rasters indicating activity of projection pools in 1ms bins, and mean phase of firing (×'s) for each pool (relative to arbitrary zero time). The abscissa shows firing time normalized by the period of oscillation (which may be converted to firing phase by multiplication by $2\pi$). Under constant input to the input projection ring, the input pools fire approximately in sequence. Two cycles of pool activity normalized by maximum firing rate for each pool are shown in left inset (for clarity, pools 1-6 are shown in the top panel and pools 7-12 are shown separately in the bottom panel); phase of background inhibition of pool 4 is shown (below) for reference. Phase-aligned average[1] of activity (right inset) showed that the firing times were relatively tight and uniform across pools: a standard deviation of 0.0945 periods, or equivalently, a spread of 1.135 pools at any instant of time.

We verified that the input projection pools fired in a phase-shifted fashion relative to one another, a property critical for accurate encoding (see Figure 2). We stimulated all pools in the input projection ring simultaneously while the input oscillation ring provided a periodic wave of background inhibition. The mean phase of firing for each pool (relative to arbitrary zero time) increased nearly linearly with pool number, thereby providing evidence for accurate, phase-based encoding (Figure 4). The firing times of all pools are shown for two cycles of background oscillatory activity (Figure 4 *left inset*). A *phase-aligned average*[1] showed that the timing was relatively tight (standard deviation 1.135 pools) and uniform across pools of neurons (Figure 4 *right inset*).

We then characterized the system's ability to correctly decode this encoding under a given circular shift. The shift was set to seven pools, mapping input pool 1 to output pool 8, and so on. Each input pool was stimulated in turn. We expected to see only the appropriately shifted output pool become highly active. In fact, not only was this pool active, but other pools around it were also active, though to a lesser extent (Figure 5A). Thus, the phase-encoded input was decoded successfully, and circularly shifted, except that the output units were broadly tuned.

To quantify the overall precision of encoding and decoding, we constructed an *input-locked average* of the tuning curves (Figure 5B): the curves were circularly shifted to the left by an amount corresponding to the stimulated input pool number, and the raw pool firing rates were averaged. If the phase-based encoding and decoding were perfect, the peak should occur at a shift of 7 pools.

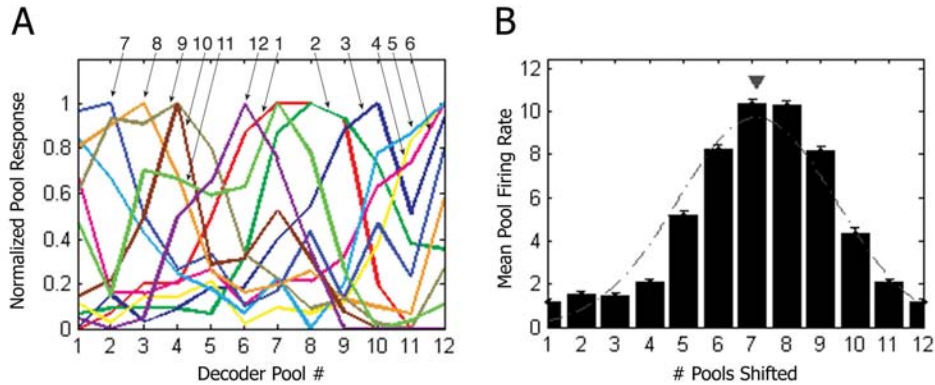

Figure 5: Decoding phase-encoded input. **A** In order to assess decoding performance under a given circular shift (here 7 pools) each input pool was stimulated in turn and activity in each output pool was recorded and averaged over 500ms. The pool's response, normalized by its maximum firing rate, is plotted for each stimulated input pool (arrows pointing to curves, color code as in Figure 4). Each input pool stimulation trial consistently resulted in peak activity in the appropriate output pool; however, adjacent pools were also active, but to a lesser extent, resulting in a broad tuning curve. **B** The best-fit Gaussian (dot-dashed grey curve, $\sigma = 2.30$ pools) to the input-locked average of the raw pool firing rates (see text for details) revealed a maximum between a shift of 7 and 8 pools (inverted grey triangle; expected peak at a shift of 7 pools).

Indeed, the highest (average) firing rate corresponded to a shift of 7 pools. However, the activity corresponding to a shift of 8 pools was nearly equal to that of 7 pools, and the best fitting Gaussian curve to the activity histogram (grey dot-dashed line) peaked at a point between pools 7 and 8 (inverted grey triangle). The standard deviation ($\sigma$) was 2.30 pools, versus the expected ideal $\sigma$ of 1.60, which corresponds to the encoding distribution ($\sigma = 1.135$ pools) convolved with itself.

## 5    Discussion

We have demonstrated a biologically plausible mechanism for the dynamic routing of information in time that obviates the need for precise gating of connections. This mechanism requires that a wave of activity propagate around pools of neurons arranged in a ring. While previous work has described traveling waves in a ring of neurons [13], and a double ring architecture (for determining head-direction) [14], our work combines these two features (twin rings with phase-shifted traveling waves) to achieve dynamic routing. These features of the model are found in the cortex: Bonhoeffer and Grinwald [15] describe iso-orientation columns in the cat visual cortex that are arranged in ring-like pinwheel patterns, with orientation tuning changing gradually around the pinwheel center. Moreover, Rubino et al. [16] have shown that coherent oscillations can propagate as waves across the cortical surface in the motor cortex of awake, behaving monkeys performing a delayed reaching task.

Our solution for CSI is also applicable to music perception. In the Western twelve-tone, equal-temperament tuning system (12-tone scale), each octave is divided into twelve logarithmically-spaced notes. Human observers are known to construct mental representations for raw notes that are invariant of the (perceived) key of the music: a note of C heard in the key of C-Major is perceptually equivalent to the note C# heard in the key of C#-Major [8,17]. In previous dynamic routing models of key invariance, the tonic—the first note of the key (e.g., C is the tonic of C-Major)—supplies the equivalent *where* information used by routing units that gate precise connections to map the raw note into a key-invariant output representation [17].

To achieve key invariance in our model, the bottom tier encodes raw note information while the top tier decodes key-invariant notes (Figure 6). The middle tier receives the tonic information and aligns the phase of the first output pool (whose invariant representation corresponds to the tonic) with the appropriate input pool (whose raw note representation corresponds to the tonic of the perceived key).

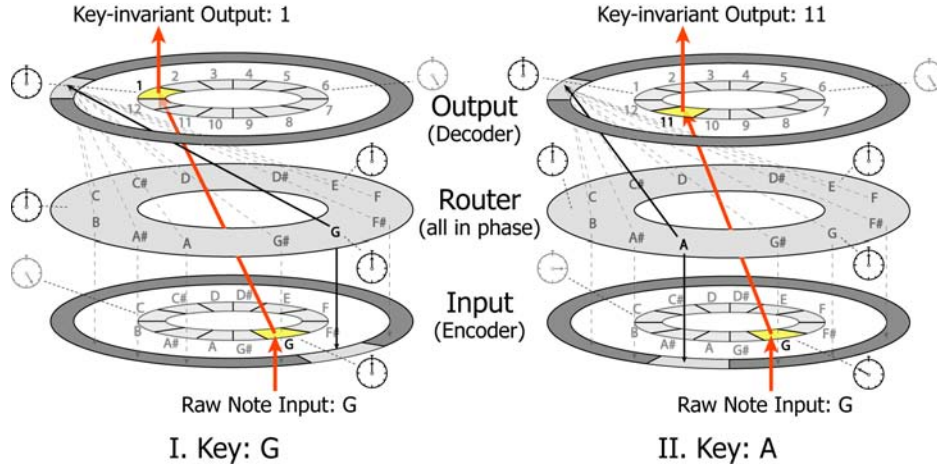

Figure 6: Phase-based dynamic routing to achieve key-invariance. The input (bottom) tier encodes raw note information, and the output (top) tier decodes key-invariant information. The routing (middle) tier sets the phase of the background wave activity in the input and output oscillation rings (dashed arrows) such that the first output pool is in phase with the input pool representing the note corresponding to the tonic. On the left, where G is the tonic, input pool G, output pool 1, and the routing tier are in phase with one another (black clocks), while input pool C and output pool 6 are in phase with one another (grey clocks). Thus, the raw note input, G, activates the invariant output 1, which corresponds to the perceived tonic invariant representation (heavy solid arrows). On the right, the same raw input note, G, is active, but the key is different and A is now the active tonic; thus the raw input, G, is now mapped to output pool 11.

The tonic information is supplied to a specific pool in the routing ring according to the perceived key. This pool projects directly down to the input pool corresponding to the tonic. This ensures that the current tonic's input pool is excitable in the same time bin as the first output pool. Each of the remaining raw input notes of the octave is mapped by time binning to the corresponding key-invariant representation in the output tier, as the phases of input pools are all shifted by the same amount. Supporting evidence for phase-based encoding of note information comes from MEG recordings in humans: the phase of the MEG signal (predominantly over right hemispheric sensor locations) tracks the note of the heard note sequence with surprising accuracy [18].

The input and output tiers' periods must be kept in lock-step, which can be accomplished through more plausible means than employed in the current implementation of this model. Here, we maintained a fixed phase shift between the input and output oscillation rings by feeding activity from the input oscillation ring to the appropriately shifted pool in the output oscillation ring. This approach allowed us to avoid difficulties achieving coherent oscillations at identical frequencies in the input and output oscillation rings. Alternatively, entrainment could be achieved even when the frequencies are not identical—a more biologically plausible scenario—if the routing ring resets the phase of the input and output rings on a cycle-by-cycle basis. Lakatos et al. [19] have shown that somatosensory inputs can reset the phase of ongoing neuronal oscillations in the primary auditory cortex (A1), which helps in the generation of a unified auditory-tactile percept (the so-called "Hearing-Hands Effect").

A simple extension to our model can reduce the number of connections below the requirements of traditional dynamic routing models. Instead of having all-to-all connections between the input and output layers, a relay layer of very few ($M \ll N$) neurons could be used to transmit the spikes form the input neurons to the output neurons (analogous to the single wire connecting encoder and decoder in Figure 1B). A small number of (or ideally even one) relay neurons suffices because encoding and decoding occur in *time*. Hence, the connections between each input pool and the relay neurons require $O(MN) \approx O(N)$ connections (as long as $M$ does not scale with $N$) and those between the relay neurons and each output pool require $O(MN) \approx O(N)$ connections as well. Thus, by removing all-to-all connectivity between the input and output units (a standard feature in traditional dynamic routing models), the number of required connections is reduced from $O(N^2)$

to O($N$). Further, by replacing the strict pool boundaries with nearest neighbor connectivity in the projection rings, the proposed model can accommodate a continuum of rotation angles.

In summary, we propose that the mechanism of dynamic routing through neuronal coherence could be a general mechanism that could be used by multiple sensory and motor modalities in the neocortex: it is particularly suitable for placing raw information in an appropriate context (defined by the routing tier).

**Acknowledgments**

DS was supported by a Stanford Graduate Fellowship and BP was supported under a National Science Foundation Graduate Research Fellowship.

**References**

[1] Olshausen B.A., Anderson C.H. & Van Essen D.C. (1993). A neurobiological model of visual attention and invariant pattern recognition based on dynamic routing of information. *Journal of Neuroscience* **13**(11):4700-4719.

[2] Wiskott L. (2004). How does our visual system achieve shift and size invariance? In *J.L. van Hemmen & T.J. Sejnowski (Eds.), 23 Problems in Systems Neuroscience*, Oxford University Press.

[3] Fukushima K., Miyake S. & Ito T. (1983). A neural network model for a mechanism of visual pattern recognition. *IEEE Transactions on Systems, Man and Cybernetics* **13**:826-834.

[4] Mel B.W., Ruderman D.L & Archie K.A. (1998). Translation invariant orientation tuning in visual "complex" cells could derive from intradendritic computations. *Journal of Neuroscience* **18**(11):4325-4334.

[5] McKone, E. & Grenfell, T. (1999). Orientation invariance in naming rotated objects: Individual differences and repetition priming. *Perception and Psychophysics*, **61**:1590-1603.

[6] Harris IM & Dux PE. (2005). Orientation-invariant object recognition: evidence from repetition blindness. *Cognition*, **95**(1):73-93.

[7] Naval Electrical Engineering Training Series (NEETS). Module 17, Radio-Frequency Communication Principles, Chapter 3, pp.32. Published online at http://www.tpub.com/content/neets/14189 (Integrated Publishing).

[8] Krumhansl C.L. (1990). Cognitive foundations of musical pitch. Oxford University Press, 1990.

[9] Fries P. (2005). A mechanism for cognitive dynamics: neuronal communication through neuronal coherence. *Trends in Cognitive Sciences* **9**(10):474-480.

[10] Buzsaki G. & Draguhn A. (2004). Neuronal Oscillations in Cortical Networks. *Science* **304**(5679):1926-1929.

[11] Sejnowski T.J. & Paulsen O. (2006). Network oscillations: Emerging computational principles. *Journal of Neuroscience* **26**(6):1673-1676.

[12] Arthur J.A. & Boahen K. (2005). Learning in Silicon: Timing is Everything. *Advances in Neural Information Processing Systems 17, B Sholkopf and Y Weiss, Eds,* MIT Press, 2006.

[13] Hahnloser R.H.R., Sarpeshkar R., Mahowald M.A., Douglas R.J., & Seung H.S. (2000). Digital selection and analogue amplification coexist in a cortex-inspired silicon circuit. *Nature* **405**:947-951.

[14] Xie X., Hahnloser R.H.R., & Seung H.S (2002). Double-ring network modeling of the head-direction system. *Phys. Rev.* **E66** 041902:1-9.

[15] Bonhoeffer K. & Grinwald A. (1991). Iso-orientation domains in cat visual cortex are arranged in pinwheel-like patterns. *Nature* **353**:426-437.

[16] Rubino D., Robbins K.A. & Hastopoulos N.G. (2006). Propagating waves mediate information transfer in the motor cortex. *Nature Neuroscience* **9**:1549-1557.

[17] Bharucha J.J. (1999). Neural nets, temporal composites and tonality. In *D. Deutsch (Ed.), The Psychology of Music (2d Ed.)* Academic Press, New York.

[18] Patel A.D. & Balaban E. (2000). Temporal patterns of human cortical activity reflect tone sequence structure. *Nature* **404**:80-84.

[19] Lakatos P., Chen C., O'Connell M., Mills A. & Schroeder C. (2007). Neuronal oscillations and multisensory interaction in primary auditory cortex. *Neuron* **53**(2):279-292.

## Footnotes

[1]The phase-aligned average was constructed by shifting the pool-activity curves by the (# of the pool) × ($\frac{1}{12}$ of the period) to align activity across pools, which was then averaged.
